# The Efficient Learning of Multiple Task Sequences

Satinder P. Singh
Department of Computer Science
University of Massachusetts
Amherst, MA 01003

## Abstract

I present a modular network architecture and a learning algorithm based on incremental dynamic programming that allows a single learning agent to learn to solve multiple Markovian decision tasks (MDTs) with significant transfer of learning across the tasks. I consider a class of MDTs, called composite tasks, formed by temporally concatenating a number of simpler, elemental MDTs. The architecture is trained on a set of composite and elemental MDTs. The temporal structure of a composite task is assumed to be unknown and the architecture learns to produce a temporal decomposition. It is shown that under certain conditions the solution of a composite MDT can be constructed by computationally inexpensive modifications of the solutions of its constituent elemental MDTs.

## 1 INTRODUCTION

Most applications of domain independent learning algorithms have focussed on learning single tasks. Building more sophisticated learning agents that operate in complex environments will require handling multiple tasks/goals (Singh, 1992). Research effort on the scaling problem has concentrated on discovering faster learning algorithms, and while that will certainly help, techniques that allow transfer of learning across tasks will be indispensable for building autonomous learning agents that have to learn to solve multiple tasks. In this paper I consider a learning agent that interacts with an external, finite-state, discrete-time, stochastic dynamical environment and faces multiple sequences of Markovian decision tasks (MDTs).

Each MDT requires the agent to execute a sequence of actions to control the environment, either to bring it to a desired state or to traverse a desired state trajectory over time. Let $S$ be the finite set of states and $A$ be the finite set of actions available to the agent.[1] At each time step $t$, the agent observes the system's current state $x_t \in S$ and executes action $a_t \in A$. As a result, the agent receives a payoff with expected value $R(x_t, a_t) \in \mathbb{R}$ and the system makes a transition to state $x_{t+1} \in S$ with probability $P_{x_t x_{t+1}}(a_t)$. The agent's goal is to learn an optimal closed loop control policy, i.e., a function assigning actions to states, that maximizes the agent's objective. The objective used in this paper is $J = \sum_{t=0}^{\infty} \gamma^t R(x_t, a_t)$, i.e., the sum of the payoffs over an infinite horizon. The discount factor, $0 \le \gamma \le 1$, allows future payoff to be weighted less than more immediate payoff. Throughout this paper, I will assume that the learning agent does not have access to a model of the environment. Reinforcement learning algorithms such as Sutton's (1988) temporal difference algorithm and Watkins's (1989) Q-learning algorithm can be used to learn to solve single MDTs (also see Barto *et al.*, 1991).

I consider compositionally-structured MDTs because they allow the possibility of sharing knowledge across the many tasks that have common subtasks. In general, there may be $n$ elemental MDTs labeled $T_1, T_2, \ldots, T_n$. *Elemental* MDTs cannot be decomposed into simpler subtasks. *Composite* MDTs, labeled $C_1, C_2, \ldots, C_m$, are produced by temporally concatenating a number of elemental MDTs. For example, $C_j = [T(j, 1) T(j, 2) \cdots T(j, k)]$ is composite task $j$ made up of $k$ elemental tasks that have to be performed in the order listed. For $1 \le i \le k$, $T(j, i) \in \{T_1, T_2, \ldots, T_n\}$ is the $i^{th}$ elemental task in the list for task $C_j$. The sequence of elemental tasks in a composite task will be referred to as the *decomposition* of the composite task; the decomposition is assumed to be unknown to the learning agent.

*Compositional learning* involves solving a composite task by learning to compose the solutions of the elemental tasks in its decomposition. It is to be emphasized that given the short-term, evaluative nature of the payoff from the environment (often the agent gets informative payoff only at the completion of the composite task), the task of discovering the decomposition of a composite task is formidable. In this paper I propose a compositional learning scheme in which separate modules learn to solve the elemental tasks, and a task-sensitive gating module solves composite tasks by learning to compose the appropriate elemental modules over time.

## 2    ELEMENTAL AND COMPOSITE TASKS

All elemental tasks are MDTs that share the the same state set $S$, action set $A$, and have the same environment dynamics. The payoff function for each elemental task $T_i$, $1 \le i \le n$, is $R_i(x, a) = \sum_{y \in S} P_{xy}(a) r_i(y) - c(x, a)$, where $r_i(y)$ is a positive reward associated with the state $y$ resulting from executing action $a$ in state $x$ for task $T_i$, and $c(x, a)$ is the positive cost of executing action $a$ in state $x$. I assume that $r_i(x) = 0$ if $x$ is not the desired final state for $T_i$. Thus, the elemental tasks share the same cost function but have their own reward functions.

A composite task is not itself an MDT because the payoff is a function of both

the state and the current elemental task, instead of the state alone. Formally, the new state set[2] for a composite task, $S'$, is formed by augmenting the elements of set $S$ by $n$ bits, one for each elemental task. For each $x' \in S'$, the *projected state* $x \in S$ is defined as the state obtained by removing the augmenting bits from $x'$. The environment dynamics and cost function, $c$, for a composite task is defined by assigning to each $x' \in S'$ and $a \in A$ the transition probabilities and cost assigned to the projected state $x \in S$ and $a \in A$. The reward function for composite task $C_j$, $r_j^c$, is defined as follows. $r_j^c(x') \geq 0$ if the following are all true: i) the projected state $x$ is the final state for some elemental task in the decomposition of $C_j$, say task $T_i$, ii) the augmenting bits of $x'$ corresponding to elemental tasks appearing before and including subtask $T_i$ in the decomposition of $C_j$ are one, and iii) the rest of the augmenting bits are zero; $r_j^c(x') = 0$ everywhere else.

## 3   COMPOSITIONAL Q-LEARNING

Following Watkins (1989), I define the Q-value, $Q(x, a)$, for $x \in S$ and $a \in A$, as the expected return on taking action $a$ in state $x$ under the condition that an optimal policy is followed thereafter. Given the Q-values, a greedy policy that in each state selects an action with the highest associated Q-value, is optimal. Q-learning works as follows. On executing action $a$ in state $x$ at time $t$, the resulting payoff and next state are used to update the estimate of the Q-value at time $t$, $\hat{Q}_t(x, a)$:

$$\hat{Q}_{t+1}(x, a) = (1.0 - \alpha_t)\hat{Q}_t(x, a) + \alpha_t[R(x, a) + \gamma \max_{a' \in A} \hat{Q}_t(y, a')], \qquad (1)$$

where $y$ is the state at time $t + 1$, and $\alpha_t$ is the value of a positive learning rate parameter at time $t$. Watkins and Dayan (1992) prove that under certain conditions on the sequence $\{\alpha_t\}$, if every state-action pair is updated infinitely often using Equation 1, $\hat{Q}_t$ converges to the true Q-values asymptotically.

Compositional Q-learning (CQ-learning) is a method for constructing the Q-values of a composite task from the Q-values of the elemental tasks in its decomposition. Let $Q_{T_i}(x, a)$ be the Q-value of $(x, a)$, $x \in S$ and $a \in A$, for elemental task $T_i$, and let $Q_{T_i}^{C_j}(x', a)$ be the Q-value of $(x', a)$, for $x' \in S'$ and $a \in A$, for task $T_i$ when performed as part of the composite task $C_j = [T(j, 1) \cdots T(j, k)]$. Assume $T_i = T(j, l)$. Note that the superscript on $Q$ refers to the task and the subscript refers to the elemental task currently being performed. The absence of a superscript implies that the task is elemental.

Consider a set of undiscounted ($\gamma = 1$) MDTs that have compositional structure and satisfy the following conditions:
(A1) Each elemental task has a single desired final state.
(A2) For all elemental and composite tasks, the expected value of undiscounted return for an optimal policy is bounded both from above and below for all states.
(A3) The cost associated with each state-action pair is independent of the task being accomplished.

(A4) For each elemental task $T_i$, the reward function $r_i$ is zero for all states except the desired final state for that task. For each composite task $C_j$, the reward function $r_j^c$ is zero for all states except *possibly* the final states of the elemental tasks in its decomposition (Section 2).

Then, for any elemental task $T_i$ and for all composite tasks $C_j$ containing elemental task $T_i$, the following holds:

$$Q_{T_i}^{C_j}(x', a) = Q_{T_i}(x, a) + K(C_j, T(j, l)), \tag{2}$$

for all $x' \in S'$ and $a \in A$, where $x \in S$ is the projected state, and $K(C_j, T(j, l))$ is a function of the composite task $C_j$ and subtask $T(j, l)$, where $T_i = T(j, l)$. Note that $K(C_j, T(j, l))$ is independent of the state and the action. Thus, given solutions of the elemental tasks, learning the solution of a composite task with $n$ elemental tasks requires learning only the values of the function $K$ for the $n$ different subtasks. A proof of Equation 2 is given in Singh (1992).

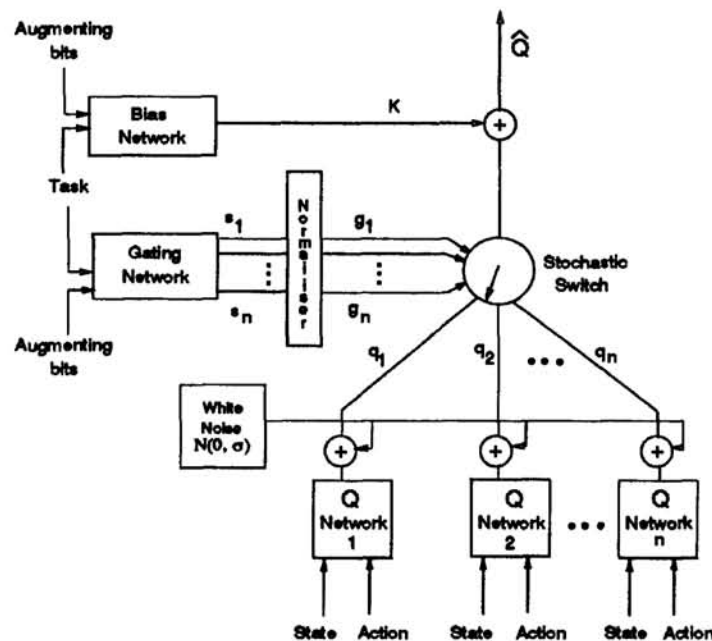

Figure 1: The CQ-Learning Architecture (CQ-L). This figure is adapted from Jacobs et al. (1991). See text for details.

Equation 2 is based on the assumption that the decomposition of the composite tasks is known. In the next Section, I present a modular architecture and learning algorithm that simultaneously discovers the decomposition of a composite task and implements Equation 2.

## 4    CQ-L: CQ-LEARNING ARCHITECTURE

Jacobs (1991) developed a modular connectionist architecture that performs task decomposition. Jacobs's gating architecture consists of several expert networks and a gating network that has an output for each expert network. The architecture has been used to learn multiple non-sequential tasks within the supervised learning

Table 1: Tasks. Tasks $T_1$, $T_2$, and $T_3$ are elemental tasks; tasks $C_1$, $C_2$, and $C_3$ are composite tasks. The last column describes the compositional structure of the tasks.

| Label | Command | Description | Decomposition |
|-------|---------|-------------|---------------|
| $T_1$ | 000001 | visit A | $T_1$ |
| $T_2$ | 000010 | visit B | $T_2$ |
| $T_3$ | 000100 | visit C | $T_3$ |
| $C_1$ | 001000 | visit A and then C | $T_1 T_3$ |
| $C_2$ | 010000 | visit B and then C | $T_2 T_3$ |
| $C_3$ | 100000 | visit A, then B and then C | $T_1 T_2 T_3$ |

paradigm. I extend the modular network architecture to a CQ-Learning architecture (Figure 1), called CQ-L, that can learn multiple compositionally-structured sequential tasks even when training information required for supervised learning is not available. CQ-L combines CQ-learning and the gating architecture to achieve transfer of learning by "sharing" the solutions of elemental tasks across multiple composite tasks. Only a very brief description of the CQ-L is provided in this paper; details are given in Singh (1992) .

In CQ-L the expert networks are Q-learning networks that learn to approximate the Q-values for the elemental tasks. The Q-networks receive as input both the current state and the current action. The gating and bias networks (Figure 1) receive as input the augmenting bits and the task command used to encode the current task being performed by the architecture. The stochastic switch in Figure 1 selects one Q-network at each time step. CQ-L's output, $\hat{Q}$, is the output of the selected Q-network added to the output of the bias network.

The learning rules used to train the network perform gradient descent in the log likelihood, $L(t)$, of generating the estimate of the desired Q-value at time $t$, denoted $D(t)$, and are given below:

$$q_j(t+1) = q_j(t) + \alpha_Q \frac{\partial \log L(t)}{\partial q_j(t)},$$

$$s_i(t+1) = s_i(t) + \alpha_g \frac{\partial \log L(t)}{\partial s_i(t)}, \text{and}$$

$$b(t+1) = b(t) + \alpha_b(D(t) - \hat{Q}(t)),$$

where $q_j$ is the output of the $j^{th}$ Q-network, $s_i$ is the $i^{th}$ output of the gating network, $b$ is the output of the bias network, and $\alpha_Q$, $\alpha_b$ and $\alpha_g$ are learning rate parameters. The backpropagation algorithm ( e.g., Rumelhart *et al.*, 1986) was used to update the weights in the networks. See Singh (1992) for details.

## 5   NAVIGATION TASK

To illustrate the utility of CQ-L, I use a navigational test bed similar to the one used by Bachrach (1991) that simulates a planar robot that can translate simultaneously

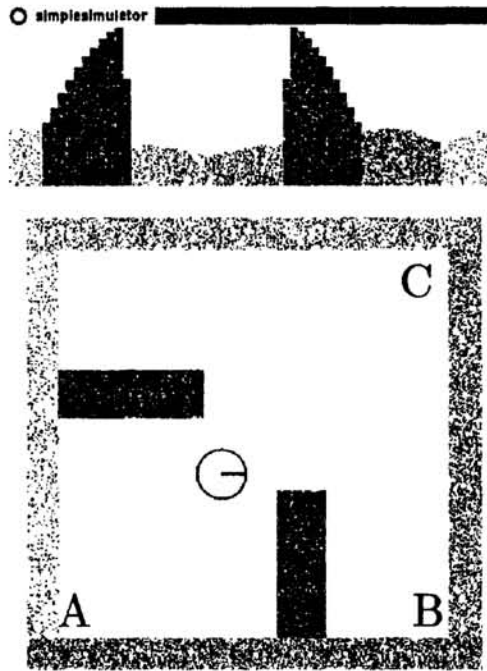

Figure 2: Navigation Testbed. See text for details.

and independently in both $x$ and $y$ directions. It can move one radius in any direction on each time step. The robot has 8 distance sensors and 8 gray-scale sensors evenly placed around its perimeter. These 16 values constitute the state vector. Figure 2 shows a display created by the navigation simulator. The bottom portion of the figure shows the robot's environment as seen from above. The upper panel shows the robot's state vector. Three different goal locations, $A$, $B$, and $C$, are marked on the test bed. The set of tasks on which the robot is trained are shown in Table 1. The elemental tasks require the robot to go to the given goal location from a random starting location in minimum time. The composite tasks require the robot to go to a goal location via a designated sequence of subgoal locations.

Task commands were represented by standard unit basis vectors (Table 1), and thus the architecture could not "parse" the task command to determine the decomposition of a composite task. Each Q-network was a feedforward connectionist network with a single hidden layer containing 128 radial basis units. The bias and gating networks were also feedforward nets with a single hidden layer containing sigmoid units. For all $x \in S \cup S'$ and $a \in A$, $c(x, a) = -0.05$. $r_i(x) = 1.0$ only if $x$ is the desired final state of elemental task $T_i$, or if $x \in S'$ is the final state of composite task $C_i$; $r_i(x) = 0.0$ in all other states. Thus, for composite tasks no intermediate payoff for successful completion of subtasks was provided.

## 6    SIMULATION RESULTS

In the simulation described below, the performance of CQ-L is compared to the performance of a "one-for-one" architecture that implements the "learn-each-task-separately" strategy. The one-for-one architecture has a pre-assigned distinct net-

work for each task, which prevents transfer of learning. Each network of the one-for-one architecture was provided with the augmented state.

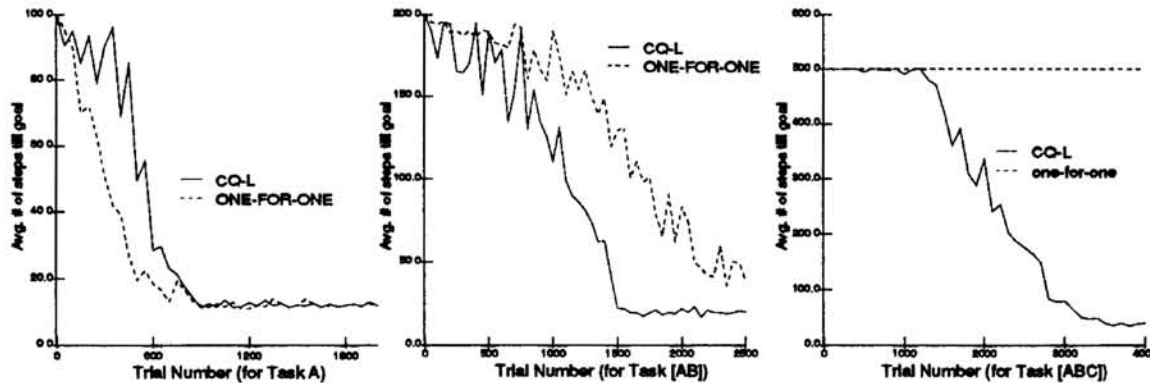

Figure 3: Learning Curves for Multiple tasks.

Both CQ-L and the one-for-one architecture were separately trained on the six tasks $T_1$, $T_2$, $T_3$, $C_1$, $C_2$, and $C_3$ until they could perform the six tasks optimally. CQ-L contained three Q-networks, and the one-for-one architecture contained six Q-networks. For each trial, the starting state of the robot and the task identity were chosen randomly. A trial ended when the robot reached the desired final state or when there was a time-out. The time-out period was 100 for the elemental tasks, 200 for $C_1$ and $C_2$, and 500 for task $C_3$. The graphs in Figure 3 show the number of actions executed per trial. Separate statistics were accumulated for each task.

The rightmost graph shows the performance of the two architectures on elemental task $T_1$. Not surprisingly, the one-for-one architecture performs better because it does not have the overhead of figuring out which Q-network to train for task $T_1$. The middle graph shows the performance on task $C_1$ and shows that the CQ-L architecture is able to perform better than the one-for-one architecture for a composite task containing just two elemental tasks. The leftmost graph shows the results for composite task $C_3$ and illustrates the main point of this paper. The one-for-one architecture is unable to learn the task, in fact it is unable to perform the task more than a couple of times due to the low probability of randomly performing the correct task sequence.

This simulation shows that CQ-L is able to learn the decomposition of a composite task and that compositional learning, due to transfer of training across tasks, can be faster than learning each composite task separately. More importantly, CQ-L is able to learn to solve composite tasks that cannot be solved using traditional schemes.

## 7   DISCUSSION

Learning to solve MDTs with large state sets is difficult due to the sparseness of the evaluative information and the low probability that a randomly selected sequence of actions will be optimal. Learning the long sequences of actions required to solve such tasks can be accelerated considerably if the agent has prior knowledge of useful subsequences. Such subsequences can be learned through experience in learning to

solve other tasks. In this paper, I define a class of MDTs, called composite MDTs, that are structured as the temporal concatenation of simpler MDTs, called elemental MDTs. I present CQ-L, an architecture that combines the Q-learning algorithm of Watkins (1989) and the modular architecture of Jacobs et al. (1991) to achieve transfer of learning by sharing the solutions of elemental tasks across multiple composite tasks. Given a set of composite and elemental MDTs, the sequence in which the learning agent receives training experiences on the different tasks determines the relative advantage of CQ-L over other architectures that learn the tasks separately. The simulation reported in Section 6 demonstrates that it is possible to train CQ-L on intermixed trials of elemental and composite tasks. Nevertheless, the ability of CQ-L to scale well to complex sets of tasks will depend on the choice of the training sequence.

## Acknowledgements

This work was supported by the Air Force Office of Scientific Research, Bolling AFB, under Grant AFOSR-89-0526 and by the National Science Foundation under Grant ECS-8912623. I am very grateful to Andrew Barto for his extensive help in formulating these ideas and preparing this paper.

## Footnotes

[1]The extension to the case where different sets of actions are available in different states is straightforward.

[2]The theory developed in this paper does not depend on the particular extension of $S$ chosen, as long as the appropriate connection between the new states and the elements of $S$ can be made.

## References

J. R. Bachrach. (1991) A connectionist learning control architecture for navigation. In R. P. Lippmann, J. E. Moody, and D. S. Touretzky, editors, *Advances in Neural Information Processing Systems 3*, pages 457–463, San Mateo, CA. Morgan Kaufmann.

A. G. Barto, S. J. Bradtke, and S. P. Singh. (1991) Real-time learning and control using asynchronous dynamic programming. Technical Report 91-57, University of Massachusetts, Amherst, MA. Submitted to *AI Journal*.

R. A. Jacobs. (1990) *Task decomposition through competition in a modular connectionist architecture*. PhD thesis, COINS dept, Univ. of Massachusetts, Amherst, Mass. U.S.A.

R. A. Jacobs, M. I. Jordan, S. J. Nowlan, and G. E. Hinton. (1991) Adaptive mixtures of local experts. *Neural Computation*, 3(1).

D. E. Rumelhart, G. E. Hinton, and R. J. Williams. (1986) Learning internal representations by error propagation. In D. E. Rumelhart and J. L. McClelland, editors, *Parallel Distributed Processing: Explorations in the Microstructure of Cognition, vol.1: Foundations*. Bradford Books/MIT Press, Cambridge, MA.

S. P. Singh. (1992) Transfer of learning by composing solutions for elemental sequential tasks. *Machine Learning*.

R. S. Sutton. (1988) Learning to predict by the methods of temporal differences. *Machine Learning*, 3:9–44.

C. J. C. H. Watkins. (1989) *Learning from Delayed Rewards*. PhD thesis, Cambridge Univ., Cambridge, England.

C. J. C. H. Watkins and P. Dayan. (1992) Q-learning. *Machine Learning*.